# Online Classification with Specificity Constraints

**Andrey Bernstein**
Department of Electrical Engineering
Technion - Israel Institute of Technology
Haifa, 32000, Israel
andreyb@tx.technion.ac.il

**Shie Mannor**
Department of Electrical Engineering
Technion - Israel Institute of Technology
Haifa, 32000, Israel
shie@ee.technion.ac.il

**Nahum Shimkin**
Department of Electrical Engineering
Technion - Israel Institute of Technology
Haifa, 32000, Israel
shimkin@ee.technion.ac.il

## Abstract

We consider the online binary classification problem, where we are given $m$ classifiers. At each stage, the classifiers map the input to the probability that the input belongs to the positive class. An online classification meta-algorithm is an algorithm that combines the outputs of the classifiers in order to attain a certain goal, without having prior knowledge on the form and statistics of the input, and without prior knowledge on the performance of the given classifiers. In this paper, we use *sensitivity* and *specificity* as the performance metrics of the meta-algorithm. In particular, our goal is to design an algorithm that satisfies the following two properties (asymptotically): (i) its average *false positive rate* (fp-rate) is under some given threshold; and (ii) its average *true positive rate* (tp-rate) is not worse than the tp-rate of *the best convex combination* of the $m$ given classifiers that satisfies fp-rate constraint, *in hindsight*. We show that this problem is in fact a special case of the *regret minimization problem with constraints*, and therefore the above goal is *not attainable*. Hence, we pose a *relaxed* goal and propose a corresponding *practical* online learning meta-algorithm that attains it. In the case of two classifiers, we show that this algorithm takes a very simple form. To our best knowledge, this is the first algorithm that addresses the problem of the average *tp-rate maximization* under average *fp-rate constraints* in the online setting.

## 1 Introduction

Consider the binary classification problem, where each input is classified into $+1$ or $-1$. A *classifier* is an algorithm which, for every input, classifies that input. In general, classifiers may produce the probability of the input to belong to class $1$. There are several metrics for the performance of the classifier in the offline setting, where a training set is given in advance. These include error (or mistake) count, true positive rate, and false positive rate; see [6] for a discussion. In particular, the *true positive rate* (tp-rate) is given by the fraction of the number of *positive* instances *correctly* classified out of the total number of the positive instances, while *false positive rate* (fp-rate) is given by the fraction of the number of *negative* instances *incorrectly* classified out of the total number of the negative instances. A receiver operating characteristics (ROC) graph then depicts different classifiers using their tp-rate on the $Y$ axis, while fp-rate on the $X$ axis (see [6]). We note that there are alternative names for these metrics in the literature. In particular, the tp-rate is also called *sensitivity*, while one minus the fp-rate is usually called *specificity*. In what follows, we prefer to use the terms tp-rate and fp-rate, as we think that they are self-explaining.

In this paper we focus on the *online* classification problem, where no training set is given in advance. We are given $m$ classifiers, which at each stage $n = 1, 2, ...$ map the input instance to the probability of the instance to belong to the positive class. An *online classification meta-algorithm* (or a *selection* algorithm) is an algorithm that combines the outputs of the given classifiers in order to attain a certain goal, without prior knowledge on the form and statistics of the input, and without prior knowledge on the performance of the given classifiers. The assumption is that the observed sequence of classification probabilities and labels comes from some unknown source and, thus, can be *arbitrary*. Therefore, it is convenient to formulate the online classification problem as a *repeated game* between an agent and some abstract opponent that stands for the collective behavior of the classifiers and the realized labels. We note that, in this formulation, we can identify the agent with a corresponding online classification meta-algorithm.

There is a rich literature that deals with the online classification problem, in the *competitive ratio* framework, such as [5, 1]. In these works, the performance guarantees are usually expressed in terms of the *mistake bound* of the algorithm. In this paper, we take a different approach. Our performance metrics will be the average tp-rate and fp-rate of the meta-algorithm, while the performance guarantees will be expressed in the *regret minimization* framework. In a seminal paper, Hannan [8] introduced *the optimal reward-in-hindsight* $r_n^*$ with respect to the empirical distribution of opponent's actions, as a performance goal of an online algorithm. In our case, $r_n^*$ is in fact the maximal tp-rate the agent could get at time $n$ by knowing the classification probabilities and actual labels beforehand, using *the best convex combination of the classifiers*. The *regret* is then defined as the difference between $r_n^*$ and the actual average tp-rate obtained by the agent. Hannan showed in [8] that there exist online algorithms whose regret converges to zero (or below) as time progresses, regardless of the opponent's actions, at $1/\sqrt{n}$ rate. Such algorithms are often called no-regret, Hannan-consistent, or universally consistent algorithms. Additional no-regret algorithms were proposed in the literature over the years, such as Blackwell's approachability-based algorithm [2] and weighted majority schemes [10, 7] (see [4] for an overview of these and other related algorithms). These algorithms can be directly applied to the problem of online classification when the goal is only to obtain no-regret with respect to the optimal tp-rate in hindsight.

However, in addition to tp-rate maximization, some performance guarantees in terms of the fp-rate are usually required. In particular, it is reasonable to require (following the Neyman-Pearson approach) that, in the long term, the average fp-rate of the agent will be below some given threshold $0 < \gamma < 1$. In this case the tp-rate can be considered as the average reward obtained by the agent, while fp-rate – as the average cost. This is in fact a special case of the *regret minimization problem with constraints* whose study was initiated by Mannor et al. in [11]. They defined the *constrained* reward-in-hindsight with respect to the empirical distribution of opponent's actions, as a performance goal of an online algorithm. This quantity is the maximal average reward the agent could get in hindsight, had he known the opponent's actions beforehand, by using any fixed (mixed) action, while satisfying the average cost constraints. The desired online algorithm then has to satisfy two requirements: (i) it should have a vanishing regret (with respect to the constrained reward-in-hindsight); and (ii) it should asymptotically satisfy the average cost constraints. It is shown in [11] that such algorithms do not exist in general. The positive result is that a relaxed goal, which is defined in terms of the *convex hull* of the constrained reward-in-hindsight over an appropriate space, is attainable. The two no-regret algorithms proposed in [11] explicitly involve either the convex hull or a *calibrated forecast* of the opponent's actions. Both of these algorithms may not be computationally feasible, since there are no efficient (polynomial time) procedures for the computation of both the convex hull and a calibrated forecast.

In this paper, we take an alternative approach to that of [11]. Instead of examining the constrained tp-rate in hindsight (or its convex hull), our starting point is the "standard" regret with respect to the optimal (*unconstrained*) tp-rate, and we consider a certain relaxation thereof. In particular, we define a simple relaxed form of the optimal tp-rate in-hindsight, by subtracting a positive constant from the latter. We then find the minimal constant needed in order to have a vanishing regret (with respect to this relaxed goal) while asymptotically satisfying the average fp-rate constraint. The motivation for this approach is as follows. We know that if the constraints are always satisfied, then the optimal tp-rate in-hindsight is attainable (using relatively simple no-regret algorithms). On the other hand, when the constraints need to be actively satisfied, we should "pay" some penalty in terms of the attainability of the tp-rate in-hindsight. In our case, we express this penalty in terms of the relaxation constant mentioned above. One of the main contributions of this paper is a *computationally*

*feasible* online algorithm, the Constrained Regret Matching (CRM) algorithm, that attains the posed performance goal. We note that although we focus in this paper on the online classification problem, our algorithm can be easily extended to the general case of regret minimization under average cost constraints.

The paper is structured as follows. In Section 2 we formally define the online classification problem and the goal of the meta-algorithm. In Section 3 we present the general problem of constrained regret minimization, and show that the online classification problem is its special case. In Section 4 we define our relaxed goal in terms of the unconstrained optimal tp-rate in-hindsight, propose the CRM algorithm, and show that it can be implemented efficiently. Section 5 discusses the special case of two classifiers and corresponding experimental results. We conclude in Section 6 with some final remarks.

## 2 Online Classification

We consider the online binary classification problem from an abstract space to $\{1, -1\}$. We are given $m$ classifiers that map an input instance to the probability that the instance belongs to the positive class. We denote by $\mathcal{A} = \{1, ...m\}$ the set of indices of the classifiers. An *online classification meta-algorithm* is an algorithm that combines the outputs of the given classifiers in order to attain a certain goal, without prior knowledge on the form and statistics of the input, and without prior knowledge on the performance of the given classifiers. In what follows, we identify the meta-algorithm with an *agent*, and use both these notions interchangeably. The time axis is discrete, with index $n = 1, 2, ....$ At stage $n$, the following events occur: (i) the input instance is presented to the classifiers (but *not* to the agent); (ii) each classifier $a \in \mathcal{A}$ outputs $f_n(a) \in [0, 1]$, which is the probability of the input to belong to class 1, and simultaneously the agent chooses a classifier $a_n$; and (iii) the correct label of the instance, $b_n \in \{1, -1\}$, is revealed.

There are several standard performance metrics of classifiers. These include *error count*, *true-positive rate* (which is also termed *recall* or *sensitivity*), and *false-positive rate* (one minus the fp-rate is usually termed *specificity*). As discussed in [6], tp-rate and fp-rate metrics have some attractive properties, such as that they are insensitive to changes in class distribution, and thus we focus on these metrics in this paper. In the online setting, no training set is given in advance, and therefore these rates have to be updated online, using the obtained data at each stage. Observe that this data is expressed in terms of the vector $z_n \triangleq \left( \{f_n(a)\}_{a \in \mathcal{A}}, b_n \right) \in [0, 1]^m \times \{-1, 1\}$. We let $r_n = r(a_n, z_n) \triangleq f_n(a_n) \, \mathbb{I} \{b_n = 1\}$ and $c_n = c(a_n, z_n) \triangleq f_n(a_n) \, \mathbb{I} \{b_n = 0\}$ denote the reward and the cost of the agent at time $n$. Note that $r_n$ is the probability that the instance with positive label at time $n$ will be classified correctly by the agent, while $c_n$ is the probability that the instance with negative label will be classified incorrectly. Then, $\bar{\beta}_{tp}(n) \triangleq \sum_{k=1}^{n} r_k / \sum_{k=1}^{n} \mathbb{I} \{b_n = 1\}$ and $\bar{\beta}_{fp}(n) \triangleq \sum_{k=1}^{n} c_k / \sum_{k=1}^{n} \mathbb{I} \{b_n = -1\}$ are the average tp-rate and fp-rate of the agent at time $n$, respectively.

Our aim is to design a meta-algorithm that will have $\bar{\beta}_{tp}(n)$ not worse than the tp-rate of *the best convex combination* of the $m$ given classifiers (in hindsight), while satisfying $\bar{\beta}_{fp}(n) \leq \gamma$, for some $0 < \gamma < 1$ (asymptotically, almost surely, for any possible sequence $z_1, z_2, ...$). In fact, this problem is a special case of the *regret minimization problem with constraints*. In the next section we thus present the general constrained regret minimization framework, and discuss its applicability to the case of online classification.

## 3 Constrained Regret Minimization

### 3.1 Model Definition

We consider the problem of an agent facing an arbitrary varying environment. We identify the environment with some abstract opponent, and therefore obtain a repeated game formulation between the agent and the opponent. The *constrained* game is defined by a tuple $(\mathcal{A}, \mathcal{Z}, r, c, \Gamma)$ where $\mathcal{A}$ denotes the *finite* set of possible actions of the agent; $\mathcal{Z}$ denotes the *compact* set of possible outcomes (or *actions*) of the environment; $r : \mathcal{A} \times \mathcal{Z} \to \mathbb{R}$ is the reward function; $c : \mathcal{A} \times \mathcal{Z} \to \mathbb{R}^\ell$ is the vector-valued cost function; and $\Gamma \subseteq \mathbb{R}^\ell$ is a convex and closed set within which the average

cost vector should lie in order to satisfy the constraints. An important special case is that of linear constraints, that is $\Gamma = \left\{ c \in \mathbb{R}^\ell : c_i \leq \gamma_i, \; i = 1, ..., \ell \right\}$ for some vector $\gamma \in \mathbb{R}^\ell$.

The time axis is discrete, with index $n = 1, 2, ....$. At time step $n$, the following events occur: (i) The agent chooses an action $a_n$, and the opponent chooses an action $z_n$, simultaneously; (ii) the agent observes $z_n$; and (iii) the agent receives a reward $r_n = r(a_n, z_n) \in \mathbb{R}$ and a cost $c_n = c(a_n, z_n) \in \mathbb{R}^\ell$. We let $\bar{r}_n \triangleq \frac{1}{n} \sum_{k=1}^n r_k$ and $\bar{c}_n \triangleq \frac{1}{n} \sum_{k=1}^n c_k$ denote the average reward and cost of the agent at time $n$, respectively. Let $\mathcal{H}_n \triangleq \mathcal{Z}^{n-1} \times \mathcal{A}^{n-1}$ denote the set of all possible histories of actions till time $n$. At time $n$, the agent chooses an action $a_n$ according to the *decision rule* $\pi_n : \mathcal{H}_n \to \Delta(\mathcal{A})$, where $\Delta(\mathcal{A})$ is the set of probability distributions over the set $\mathcal{A}$. The collection $\pi = \{\pi_n\}_{n=1}^\infty$ is the *strategy* of the agent. That is, at each time step, a strategy prescribes some *mixed* action $p \in \Delta(\mathcal{A})$, based on the observed history. A strategy for the opponent is defined similarly. We denote the mixed action of the opponent by $q \in \Delta(\mathcal{Z})$, which is the probability *density* over $\mathcal{Z}$.

In what follows, we will use the shorthand notation $r(p, q) \triangleq \sum_{a \in \mathcal{A}} p(a) \int_{z \in \mathcal{Z}} q(z) r(a, z)$ for the expected reward under mixed actions $p \in \Delta(\mathcal{A})$ and $q \in \Delta(\mathcal{Z})$. The notation $r(a, q), c(p, q), c(p, z), c(a, q)$ will be interpreted similarly. We make the following assumption that the agent can satisfy the constraints *in expectation* against any mixed action of the opponent.

**Assumption 3.1** (Satisfiability of Constraints)**.** *For every $q \in \Delta(\mathcal{Z})$, there exists $p \in \Delta(\mathcal{A})$, such that $c(p, q) \in \Gamma$.*

Assumption 3.1 is essential, since otherwise the opponent can violate the average-cost constraints simply by playing the corresponding stationary strategy $q$.

Let $\bar{q}_n(z) \triangleq \sum_{k=1}^n \delta\{z - z_k\}/n$ denote the *empirical density* of the opponent's actions at time $n$, so that $\bar{q}_n \in \Delta(\mathcal{Z})$. The optimal reward-in-hindsight is then given by

$$r_n^*(z_1, ..., z_n) \triangleq \frac{1}{n} \max_{a \in \mathcal{A}} \sum_{k=1}^n r(a, z_k) = \max_{a \in \mathcal{A}} \int_{z \in \mathcal{Z}} r(a, z) \frac{1}{n} \sum_{k=1}^n \delta\{z - z_k\} = \max_{a \in \mathcal{A}} r(a, \bar{q}_n),$$

implying that $r_n^* = r^*(\bar{q}_n)$. In what follows, we will use the term "reward envelope" in order to refer to functions $\rho : \Delta(\mathcal{Z}) \to \mathbb{R}$. The simplest reward envelope is the (unconstrained) *best-response* envelope (BE) $\rho = r^*$. The $n$-stage *regret* of the algorithm (with respect to the BE) is then $r^*(\bar{q}_n) - \bar{r}_n$. The *no-regret* algorithm must ensure that the regret vanishes as $n \to \infty$ regardless of the opponent's actions. However, in our case, in addition to vanishing regret, we need to satisfy the cost constraints. Obviously, the BE need not be attainable in the presence of constraints, and therefore other reward envelopes should be considered. Hence, we use the following definition (introduced in [11]) in order to assess the online performance of the agent.

**Definition 3.1** (Attainability and No-Regret)**.** *A reward envelope $\rho : \Delta(\mathcal{Z}) \to \mathbb{R}$ is $\Gamma$-attainable if there exists a strategy $\pi$ for the agent such that, almost surely, (i) $\limsup_{n \to \infty} (\rho(\bar{q}_n) - \bar{r}_n) \leq 0$, and (ii) $\lim_{n \to \infty} d(\bar{c}_n, \Gamma) = 0$, for every strategy of the opponent. Here, $d(\cdot, \Gamma)$ is Euclidean set-to-point distance. Such a strategy $\pi$ is called constrained no-regret strategy with respect to $\rho$.*

A natural extension of the BE to the constrained setting was defined in [11], by noting that if the agent knew in advance that the empirical distribution of the opponents actions is $\bar{q}_n = q$, he could choose the constrained best response mixed action $p$, which is a solution of the corresponding optimization problem:

$$r_\Gamma^*(q) \triangleq \max_{p \in \Delta(\mathcal{A})} \{r(p, q) : \text{ so that } c(p, q) \in \Gamma\}. \tag{1}$$

We refer to $r_\Gamma^*$ as the constrained best-response envelope (CBE).

The first positive result that appeared in the literature was that of Shimkin [12], which showed that the *value* $v_\Gamma \triangleq \min_{q \in \Delta(\mathcal{Z})} r_\Gamma^*(q)$ of the constrained game is attainable by the agent. The algorithm which attains the value is based on Blackwell's approachability theory [3], and is computationally efficient provided that $v_\Gamma$ can be computed offline. Unfortunately, it was shown in [11] that $r_\Gamma^*(q)$ itself is *not* attainable in general. However, the (lower) *convex hull* of $r_\Gamma^*(q)$, conv $(r_\Gamma^*)$, is attainable[1]. Two no-regret algorithms with respect to conv $(r_\Gamma^*)$ are suggested in [11]. To our best knowledge,

these algorithms are inefficient (i.e., not polynomial); these are the only existing constrained no-regret algorithms in the literature.

It should be noted that the problem that is considered here *can not* be formulated as an instance of *online convex optimization* [13, 9] – see [11] for a discussion on this issue.

## 3.2 Application to the Online Classification Problem

For the model described in Section 2, $\mathcal{A} = \{1, ..., m\}$ denotes the set of possible classifiers and $\mathcal{Z}$ denotes the set of possible outputs of the classifiers and the true labels, that is: $z = \left(\{f(a)\}_{a \in \mathcal{A}}, b\right) \in [0, 1]^m \times \{-1, 1\} \triangleq \mathcal{Z}$. The reward at time $n$ is $r_n = r(a_n, z_n) = f_n(a_n) \, \mathbb{I} \{b_n = 1\}$ and the cost is $c_n = c(a_n, z_n) = f_n(a_n) \, \mathbb{I} \{b_n = -1\}$. Note that in this case, the mixed action of the opponent $q \in \Delta(\mathcal{Z})$ is $q(f, b) = q(f|b)q(b)$, where $q(f|b)$ is the conditional density of the predictions of the classifiers and $q(b)$ is the probability of the label $b$. It is easy to check that

$$r(p, q) = q(1) \sum_{a \in \mathcal{A}} p(a)\beta_{tp}(q; a), \tag{2}$$

where $\beta_{tp}(q; a) \triangleq \int_f f(a)q(f|1)$ is the tp-rate of classifier $a$ under distribution $q$. Regarding the cost, the goal is to keep it under a given threshold $0 < \gamma < 1$. Since the regret minimization framework requires *additive* rewards and costs, we define the following modified cost function: $c_\gamma(a, z) \triangleq c(a, z) - \gamma \, \mathbb{I} \{b = -1\}$, and similarly to the reward above, we have that

$$c_\gamma(p, q) = q(-1) \left( \sum_{a \in \mathcal{A}} p(a)\beta_{fp}(q; a) - \gamma \right), \tag{3}$$

where $\beta_{fp}(q; a) \triangleq \int_f q(f| - 1)f(a)$ is the fp-rate of classifier $a$ under distribution $q$. We note that keeping the average fp-rate of the agent $\bar{\beta}_{fp}(n) \leq \gamma$ is equivalent to keeping $(1/n) \sum_{k=1}^n c_\gamma(a_k, z_k) \leq 0$.

Since our goal is to keep the fp-rate below $\gamma$, some assumption on classifiers should be imposed in order to satisfy Assumption 3.1. We assume here that the classifiers' single-stage false-positive probability is such that it allows satisfying the constraint. In particular, we redefine[2] $\mathcal{Z} \triangleq \{z = (f, b) \in [0, 1]^m \times \{-1, 1\} : \text{if } b = -1, f(a) \leq \gamma_a\}$, where $0 \leq \gamma_a \leq 1$, and there exists $a^*$ such that $\gamma_{a^*} < \gamma$. Under this assumption, it is clear that for every $q \in \Delta(\mathcal{Z})$, there exists $p \in \Delta(\mathcal{A})$, such that $c_\gamma(p, q) \leq 0$; in fact this $p$ is the probability mass concentrated on $a^*$. If additional prior information is available on the single-stage performance of the given classifiers, this may be usefully used to further restrict the set $\mathcal{Z}$. For example, we can also restrict $z = (f, 1)$ by $f(a) \geq \lambda_a$ for some $0 < \lambda_a < 1$. Such additional restrictions will generally contribute to reducing the value of the optimal relaxation parameter $\alpha^*$ (see (7) below). This effect will be explicitly demonstrated in Section 5.

We proceed to compute the BE and CBE. Using (2), the BE is

$$r^*(q) \triangleq \max_{a \in \mathcal{A}} r(a, q) = q(1) \max_{a \in \{1, ..., m\}} \{\beta_{tp}(q; a)\} \triangleq q(1)\beta^*(q), \tag{4}$$

where $\beta^*(q)$ is the optimal (unconstrained) tp-rate in hindsight under distribution $q$. Now, using (1), (2), and (3) we have that $r_\gamma^*(q) = q(1)\beta_\gamma^*(q)$, where

$$\beta_\gamma^*(q) \triangleq \max_{p \in \Delta(\mathcal{A})} \left\{ \sum_{a \in \mathcal{A}} p(a)\beta_{tp}(q; a) : \text{ so that } \sum_{a \in \mathcal{A}} p(a)\beta_{fp}(q; a) \leq \gamma \right\}, \tag{5}$$

is the optimal constrained tp-rate in hindsight under distribution $q$. Finally, note that the value of the constrained game $v_\gamma \triangleq \min_{q \in \Delta(\mathcal{Z})} r_\gamma^*(q) = 0$ in this case.

As a consequence of this formulation, the algorithms proposed in [11] can be in principle used in order to attain *the convex hull* of $r_\gamma^*$. However, given the implementation difficulties associated with these algorithms, we are motivated to examine more carefully the problem of regret minimization with constraints and provide more practical no-regret algorithms with formal guarantees.

# 4 Constrained Regret Matching

We next define a relaxed reward envelope for the online classification problem. The proposed is in fact applicable to the problem of constrained regret minimization in general. However, due to space limitation, we present it directly for our classification problem.

Our starting point here in defining an attainable reward envelope will be the BE $r^*(q) = q(1)\beta^*(q)$. Clearly, $r^*$ is in general not attainable in the presence of fp-constraints, and we thus consider a relaxed version thereof. For $\alpha \geq 0$, set $r^*_\alpha(q) \triangleq q(1)(\beta^*(q) - \alpha)$. Obviously, $r^*_\alpha$ is a convex function, and we can always pick $\alpha \geq 0$ large enough, such that $r^*_\alpha$ is attainable. Furthermore, recall that the value $v_\gamma$ of the constrained game is attainable by the agent. Observe that, generally, $r^*_\alpha(q)$ can be smaller than $v_\gamma = 0$. We thus introduce the following modification:

$$r^{\text{SR}}_\alpha(q) \triangleq q(1)\max\{0, \beta^*(q) - \alpha\}. \tag{6}$$

We refer to $r^{\text{SR}}_\alpha$ as the scalar-relaxed best-response envelope (SR-BE). Now, let[3]

$$\alpha^* \triangleq \max_{q \in \Delta(\mathcal{Z})} \left(\beta^*(q) - \beta^*_\gamma(q)\right). \tag{7}$$

We note that $r^{\text{SR}}_{\alpha^*}(q)$ is *strictly* above 0 at some point, unless the game is in some sense trivial (see the supplementary material for a proof). According to Definition 3.1, we are seeking for a strategy $\pi$ that is: (i) an $\alpha$-relaxed *no-regret* strategy for the average reward, and (ii) ensures that the cost constraints are asymptotically satisfied. Thus, at each time step, we need to balance between the need of maximizing the average tp-rate and satisfying the average fp-rate constraint. Below we propose an algorithm which solves this trade-off for $\alpha \geq \alpha^*$.

We introduce some further notation. Let

$$R^\alpha_k(a) \triangleq [f_k(a) - f_k(a_k) - \alpha] \, \mathbb{I}\{b_k = 1\}, \quad a \in \mathcal{A}, \quad L_k \triangleq c_\gamma(a_k, z_k), \tag{8}$$

denote the *instantaneous $\alpha$-regret* and the *instantaneous constraint violation* (respectively) at time $k$. We have that the average $\alpha$-regret and constraints violation at time $n$ are

$$\overline{R}^\alpha_n(a) = \bar{q}_n(1)\left[\beta_{tp}(\bar{q}_n; a) - \bar{\beta}_{tp}(n) - \alpha\right], \quad a \in \mathcal{A}; \quad \overline{L}_n = \bar{q}_n(0)[\bar{\beta}_{fp}(n) - \gamma]. \tag{9}$$

Using this notation, the Constrained Regret Matching (CRM) algorithm is given in Algorithm 1. We then have the following result.

**Theorem 4.1.** *Suppose that the CRM algorithm is applied with parameter $\alpha \geq \alpha^*$, where $\alpha^*$ is given in (7). Then, under Assumption 3.1, it attains $r^{\text{SR}}_\alpha$ (6) in the sense of Definition 3.1. That is, (i) $\liminf_{n \to \infty} \left(\bar{\beta}_{tp}(n) - \max\{0, \max_{a \in \mathcal{A}} \beta_{tp}(\bar{q}_n; a) - \alpha\}\right) \geq 0$, and (ii) $\limsup_{n \to \infty} \bar{\beta}_{fp}(n) \leq 0$, for every strategy of the opponent, almost surely.*

The proof of this Theorem is based on Blackwell's approachability theory [3], and is given in the supplementary material. We note that the mixed action required by the CRM algorithm always exists provided that $\alpha \geq \alpha^*$. It can be easily shown (see the supplementary material) that whenever $\sum_{a \in \mathcal{A}} \left[\overline{R}^\alpha_{n-1}(a)\right]_+ > 0$, this action can be computed by solving the following linear program:

$$\min_{p \in B_n} \sum_{a \in \mathcal{A}: p^\alpha_n(a) > p(a)} (p^\alpha_n(a) - p(a)), \tag{10}$$

where $B_n \triangleq \left\{p \in \Delta(\mathcal{A}) : \left[\overline{L}_{n-1}\right]_+ \left(\sum_{a' \in \mathcal{A}} p(a')f(a') - \gamma\right) \leq 0, \ \forall z = (f, -1) \in \mathcal{Z}\right\}$ and $p^\alpha_n(a) = \left[\overline{R}^\alpha_{n-1}(a)\right]_+ / \sum_{a' \in \mathcal{A}} \left[\overline{R}^\alpha_{n-1}(a')\right]_+$ is the $\alpha$-regret matching strategy. Note also that when the average constraints violation $\overline{L}_{n-1}$ is non-positive, the minimum in (10) is obtained by $p = p^\alpha_n$. Finally, when $\sum_{a \in \mathcal{A}} \left[\overline{R}^\alpha_{n-1}(a)\right]_+ = 0$, any action $p \in B_n$ can be chosen. It is worth mentioning that our algorithm, and in particular the program (10), *can not* be formulated in the Online Convex Programming (OCP) framework [13, 9], since the equivalent reward functions in our case are trajectory-dependent, while in the OCP it is assumed that these functions are arbitrary, but *fixed* (i.e., they should not depend on the agent's actions).

**Algorithm 1** CRM Algorithm

---

**Parameter:** $\alpha \geq 0$.

**Initialization:** At time $n = 0$ use arbitrary action $a_0$.

**At times** $n = 1, 2, ...$ find a mixed action $p \in \Delta(\mathcal{A})$ such that

$$
\begin{cases}
\sum_{a \in \mathcal{A}} \left[ \overline{R}_{n-1}^{\alpha}(a) \right]_{+} \left( f(a) - \sum_{a' \in \mathcal{A}} p(a') f(a') - \alpha \right) \leq 0, & \forall z = (f, 1) \in \mathcal{Z}, \\
\left[ \overline{L}_{n-1} \right]_{+} \left( \sum_{a' \in \mathcal{A}} p(a') f(a') - \gamma \right) \leq 0, & \forall z = (f, -1) \in \mathcal{Z},
\end{cases}
\tag{11}
$$

where $\overline{R}_n^{\alpha}(a)$ and $\overline{L}_{n,i}$ are given in (9). Draw classifier $a_n$ from $p$.

---

**Remark.** In practice, it may be possible to attain $r_{\alpha}^{\text{SR}}$ with $\alpha < \alpha^*$ if the opponent is not entirely adversarial. In order to capitalize on this possibility, an adaptive algorithm can be used that adjusts the value of $\alpha$ online. The idea is to start from some small initial value $\alpha_0 \geq 0$ (possibly $\alpha_0 = 0$). At each time step $n$, we would like to use a parameter $\alpha = \alpha_n$ for which inequality (11) can be satisfied. This inequality is always satisfied when $\alpha \geq \alpha^*$. If however $\alpha < \alpha^*$, the inequality may or may not be satisfied. In the latter case, $\alpha$ can be increased so that the condition is satisfied. In addition, once in a while, $\alpha$ can be reset to $\alpha_0$, in order to obtain better results. In the supplementary material we further discuss the adaptive scheme, and prove a convergence rate for it. We note that the adaptive scheme does not require the computation of the optimal $\alpha^*$, as it discovers it online.

## 5 The Special Case of Two Classifiers

If $m = 2$, we can obtain explicit expressions for the reward envelopes and for the algorithm. In particular, we have two classifiers, and we assume that the outputs of these classifiers lie in the set $\mathcal{Z} \triangleq \left\{ z \in (f, b) \in [0, 1]^2 \times \{-1, 1\} : \text{if } b = -1, f(1) \leq \gamma_1, f(2) \leq \gamma_2; \text{if } b = 1, f(2) \geq \lambda \right\}$ such that $\gamma_1 > \gamma$, $\gamma_2 < \gamma$, and $\lambda \geq 0$. Observe that under this assumption, classifier 2 has one-stage performance guarantees that will allow to obtain better guarantees of the meta-algorithm. By computing explicitly the CBE, we obtain

$$
r_{\gamma}^*(q) = q(1) \begin{cases}
\frac{\gamma - \beta_{fp}(q;2)}{\beta_{fp}(q;1) - \beta_{fp}(q;2)} \beta_{tp}(q;1) + \frac{\beta_{fp}(q;1) - \gamma}{\beta_{fp}(q;1) - \beta_{fp}(q;2)} \beta_{tp}(q;2), & \text{if } \beta_{tp}(q;1) > \beta_{tp}(q;2) \\
 & \text{and } \beta_{fp}(q;1) > \gamma, \\
\beta_{tp}(q;1), & \text{if } \beta_{tp}(q;1) > \beta_{tp}(q;2) \\
 & \text{and } \beta_{fp}(q;1) \leq \gamma, \\
\beta_{tp}(q;2), & \text{otherwise.}
\end{cases}
$$

Therefore, the relaxation parameter is

$$
\alpha = \max_{q: \, \beta_{tp}(q;1) > \beta_{tp}(q;2), \beta_{fp}(q;1) > \gamma} \left\{ \frac{\beta_{tp}(q;1) - \beta_{tp}(q;2)}{\beta_{fp}(q;1) - \beta_{fp}(q;2)} \left( \beta_{fp}(q;1) - \gamma \right) \right\} = \frac{(1 - \lambda)(\gamma_1 - \gamma)}{\gamma_1 - \gamma_2}.
$$

Finally, it is easy to check using (10) that Algorithm 1 reduces in this case to the following simple rule: (i) if $\sum_{a \in \mathcal{A}} \left[ \overline{R}_{n-1}^{\alpha}(a) \right]_{+} > 0$, choose $p(1) = \min \left\{ p_n^{\alpha}(1), \frac{\gamma - \gamma_2}{\gamma_1 - \gamma_2} \right\}$, where $p_n^{\alpha}(1) = \left[ \overline{R}_{n-1}^{\alpha}(1) \right]_{+} / \sum_{a \in \mathcal{A}} \left[ \overline{R}_{n-1}^{\alpha}(a) \right]_{+}$ denotes the $\alpha$-regret matching strategy; (ii) otherwise, choose arbitrary action with $p(1) \leq \frac{\gamma - \gamma_2}{\gamma_1 - \gamma_2}$.

We simulated the CRM algorithm with the following parameters: $\gamma = 0.3, \gamma_1 = 0.4, \gamma_2 = 0.2, \lambda = 0.7$. This gives the relaxation parameter of $\alpha = 0.15$. Half of the input instances were positives and the other half were negatives (on average). The time was divided into episodes with exponentially growing lengths. At each *odd* episode, both classifiers had similar tp-rate and both of them satisfied the constraints, while in each *even* episode, classifier 1 was perfect in positives' classification, but did not satisfy the constraints. The results are shown in Figure 1. We compared the performance of the CRM algorithm to a simple unconstrained no-regret algorithm that treats both the true-positive and false positive probabilities similarly, but with different weight. In particular, the reward at stage $n$ of this algorithm is $g_n(w) = f_n(a_n) \mathbb{I}\{b_n = 1\} - w f_n(a_n) \mathbb{I}\{b_n = -1\}$ for some weight parameter

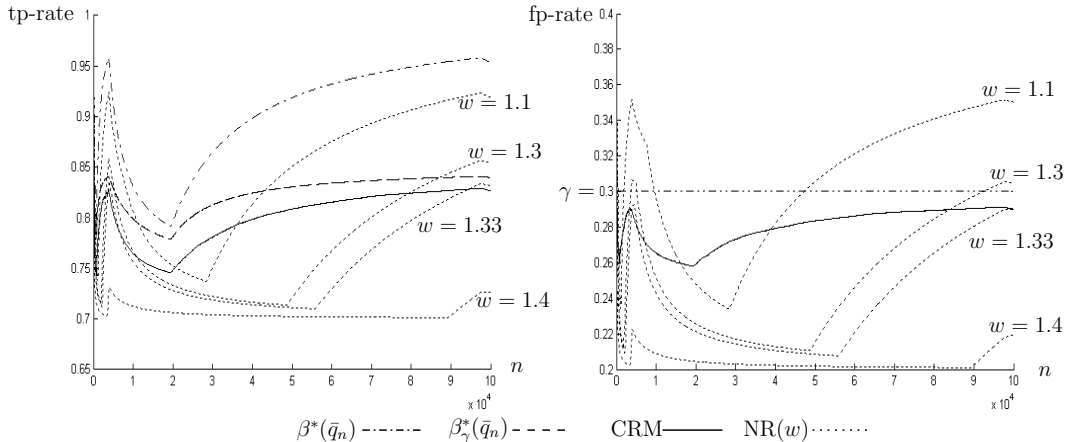

Figure 1: Experimental results for the case of two classifiers.

$w \geq 0$. Given $w$, this is simply a no-regret algorithm with respect to $g_n(w)$. When $w = 0$, the algorithm performs tp-rate maximization, while if $w$ is large, it performs fp-rate minimization. We call this algorithm NR$(w)$. As can be seen from Figure 1, the CRM algorithm outperforms NR$(w)$ for any *fixed* parameter $w$. For $w = 1.1$, NR$(w)$ has a better tp-rate, but the fp-rate constraint is violated most of the time. For $w = 1.4$, the constraints are always satisfied, but the tp-rate is always dominated by that of the CRM algorithm. For $w = 1.3, 1.33$ it can be seen that the constraints are satisfied (or almost satisfied), but the tp-rate is usually dominated by that of the CRM algorithm.

## 6   Conclusion

We studied regret minimization with average-cost constraints, with the focus on computationally feasible algorithm for the special case of online classification problem with specificity constraints. We defined a relaxed version of the best-response reward envelope and showed that it can be attained by the agent while satisfying the constraints, provided that the relaxation parameter is above a certain threshold. A polynomial no-regret algorithm was provided. This algorithm generally solves a linear program at each time step, while in some special case the algorithm's mixed action reduces to the simple $\alpha$-regret matching strategy. To the best of our knowledge, this is the first algorithm that addresses the problem of the average tp-rate maximization under average fp-rate constraints in the online setting. In addition, an adaptive scheme that adapts the relaxation parameter online was briefly discussed. Finally, the special case of two classifiers was discussed, and the experimental results for this case show that our algorithm outperforms a simple no-regret algorithm which takes as the reward function a weighted sum of the tp-rate and fp-rate.

Some remarks about our algorithm and results follow. First, the guaranteed convergence rate of the algorithm is of $O(1/\sqrt{n})$ since it is based on Blackwell's approachability theorem[4]. Second, additional constraints can be easily incorporated in the presented framework, since the general regret minimization framework assumes a *vector* of constraints. Third, it seems that there is an inherent trade-off between complexity and performance in the studied problem. In particular, in case of a single constraint, the maximal attainable relaxed goal is the convex hull of the CBE (see [11]) but no polynomial algorithms are known that attain this goal. Our results show that, by further relaxing the goal, it is possible to devise attaining polynomial algorithms. Finally, we note that the assumption on the single-stage fp-rates of the classifiers can be weakened by assuming that, in each *sufficiently large period of time*, the *average* fp-rate of each classifier $a$ is bounded by $\gamma_a$. Our approach and results can be then extended to this case, by treating each such period as a single stage.

## Footnotes

[1]The (lower) convex hull of a function $f : X \to \mathbb{R}$ is the largest *convex* function which is nowhere larger than $f$.

[2]This assumption can always be satisfied by adding a *fictitious* classifier $a_0$ that always outputs a fixed $f(a_0) < \gamma$, irrespectively of the data. However, such an addition might adversely affect the value of the optimal relaxation parameter $\alpha^*$ (see (7) below), and should be avoided if possible.

[3]In general, the parameter $\alpha^*$ may be difficult to compute analytically. See the supplementary material for a discussion on computational aspects. Also, in the supplementary material we propose an adaptive algorithm which avoids this computation (see a remark at the end of Section 4). Finally, in Section 5 we show that in the case of two classifiers this computation is trivial.

[4]A straightforward application of this theorem also gives $\sqrt{m}$ dependence of the rate on the number of classifiers. We note that it is possible to improve the dependence to $\log(m)$ by using a potential based Blackwell's approachability strategy (see for example [4], Chapter 7.8)

# References

[1] Y. Amit, S. Shalev-Shwartz, and Y. Singer. Online classification for complex problems using simultaneous projections. In *NIPS 2006*.

[2] D. Blackwell. Controlled random walks. In *Proceedings of the International Congress of Mathematicians*, volume III, pages 335–338, 1954.

[3] D. Blackwell. An analog of the minimax theorem for vector payoffs. *Pacific Journal of Mathematics*, 6:1–8, 1956.

[4] N. Cesa-Bianchi and G. Lugosi. *Prediction, Learning, and Games*. Cambridge University Press, New York, NY, USA, 2006.

[5] K. Crammer, O. Dekel, J. Keshet, S. Shalev-Shwartz, and Y. Singer. Online passive-aggressive algorithms. *Journal of Machine Learning Research*, 7:551–585, 2006.

[6] T. Fawcett. An introduction to ROC analysis. *Pattern Recognition Letters*, 27(8):861–874, 2006.

[7] Y. Freund and R.E. Schapire. Adaptive game playing using multiplicative weights. *Games and Economic Behavior*, 29(12):79–103, 1999.

[8] J. Hannan. Approximation to Bayes risk in repeated play. *Contributions to the Theory of Games*, 3:97–139, 1957.

[9] E. Hazan, A. Agarwal, and S. Kale. Logarithmic regret algorithms for online convex optimization. *Machine Learning*, 69(2-3):169–192, 2007.

[10] N. Littlestone and M. K. Warmuth. The weighted majority algorithm. *Information and Computation*, 108(2):212–261, 1994.

[11] S. Mannor, J. N. Tsitsiklis, and J. Y. Yu. Online learning with sample path constraints. *Journal of Machine Learning Research*, 10:569–590, 2009.

[12] N. Shimkin. Stochastic games with average cost constraints. *Annals of the International Society of Dynamic Games, Vol. 1: Advances in Dynamic Games and Applications*, 1994.

[13] M. Zinkevich. Online convex programming and generalized infinitesimal gradient ascent. In *Proceedings of the 20th International Conference on Machine Learning (ICML '03)*, pages 928–936, 2003.

